# Forecasting Demand for Electric Power

Jen–Lun Yuan and Terrence L. Fine
School of Electrical Engineering
Cornell University
Ithaca, NY 14853

## Abstract

We are developing a forecaster for daily extremes of demand for electric power encountered in the service area of a large midwestern utility and using this application as a testbed for approaches to input dimension reduction and decomposition of network training. Projection pursuit regression representations and the ability of algorithms like SIR to quickly find reasonable weighting vectors enable us to confront the vexing architecture selection problem by reducing high–dimensional gradient searchs to fitting single-input single-output (SISO) subnets. We introduce dimension reduction algorithms, to select features or relevant subsets of a set of many variables, based on minimizing an index of level–set dispersions (closely related to a projection index and to SIR), and combine them with backfitting to implement a neural network version of projection pursuit. The performance achieved by our approach, when trained on 1989, 1990 data and tested on 1991 data, is comparable to that achieved in our earlier study of backpropagation trained networks.

## 1    Introduction

Our work has the intertwined goals of:

(i) contributing to the improvement of the short–term electrical load (demand) forecasts used by electric utilities to buy and sell power and ensure that they can meet demand;

(ii) reducing the computational burden entailed in gradient–based training of neural networks and thereby enabling the exploration of architectures;

(iii) improving prospects for good statistical generalization by use of rational methods for reducing complexity through the identification of good small subsets of variables drawn from a large set of candidate predictor variables (feature selection);

(iv) benchmarking backpropagation and neural networks as an approach to the applied problem of load forecasting.

Our efforts proceed in the context of a problem suggested by the operational needs of a particular electric utility to make daily forecasts of short–term load or demand. Forecasts are made at midday (1 p.m.) on a weekday $t$ ( Monday - Thursday), for the next evening peak $e(t)$ (occuring usually about 8 p.m. in the winter), the daily minimum $d(t + 1)$ (occuring about 4 a.m. the next morning ) and the morning peak $m(t + 1)$ (about noon ). In addition, on Friday we are to forecast these three variables for the weekend through the Monday morning peak. These daily extremes of demand are illustrated in an excerpt from our hourly load data plotted in Figure 1.

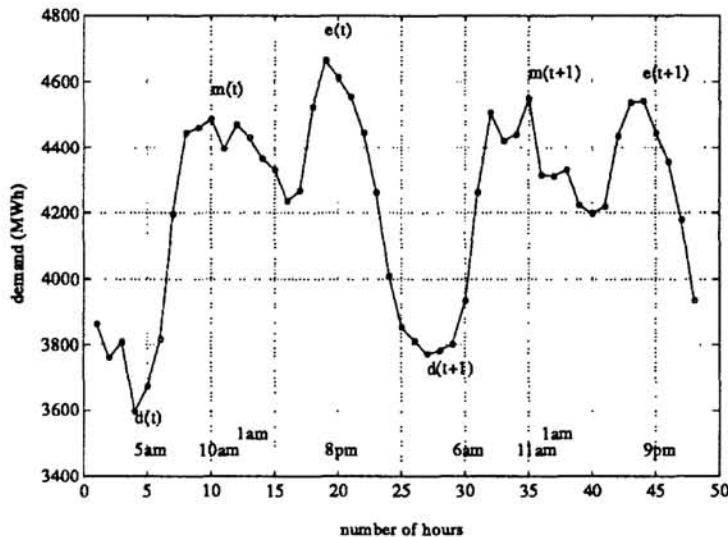

Figure 1: Hourly demand for two consecutive days showing the intended forecasting variables.

In this paper, we focus on forecasting these extremal demands up to three days ahead ( e.g. forecasting on Fridays). Neural network–based forecasters are developed which parallel the recently proposed method of slicing inverse regression (SIR) (Li [1991]) and then use backfitting (Hastie and Tibshirani [1990]) to implement a training algorithm for a projection pursuit model (Friedman [1987], Huber [1985]) that can be implemented with a single hidden layer network. Our data consists of hourly integrated system demand (MWH) and hourly temperatures measured at three cities in the service area of a large midwestern utility during 1989-91. We use 1989 and 1990 for a training set and test over the whole of 1991, with the exception of holidays that occur so infrequently that we have no training base.

## 2    Baseline Performance

### 2.1    Previous Work on Load Forecasting

Since demand is a process which does not have a known physical or mathematical model, we do not know the best achievable forecasting performance, and we are led to making comparisons with methods and results reported elsewhere. There is a substantial literature on short–term load forecasting, with Gross et al. [1987] and Willis et al. [1984] providing good reviews of approaches based upon such statistical methods as linear least squares regression and Box–Jenkins and ARMAX time series models. Many utilities rely upon the seemingly seat–of–the–pants estimates produced by individuals who have been long employed at this task and who extrapolate from a large historical data base. In the past few years there have been several efforts to employ neural networks trained through backpropagation. In two such recent studies conducted at the Univ. of Washington an average peak error of 2.04% was reported by Damborg et al. [1990] and an hourly load error of about 2.2% was given by Connor et al. [1991]. However, the accuracies reported in the literature are difficult to compare with since utilities are exposed to different operating conditions (e.g., weather, residential/industrial balance). To provide a benchmark for the error performance achieved by our method, we evaluated three basic forecasting models on our data. These methods are based on a pair of features made plausible by the scatter plots shown in Figure 2.

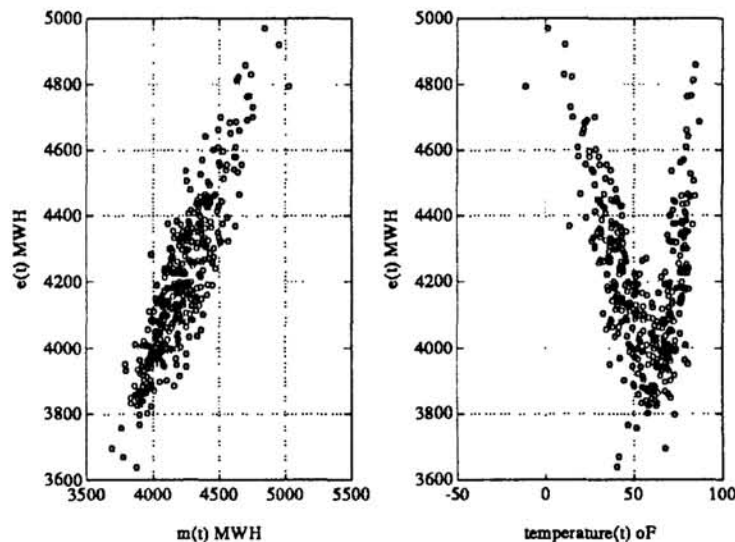

Figure 2: Evening peaks (Tue.-Fri.,1989-90) vs. morning peaks and temperatures.

### 2.2    Feature Selection and Homogeneous Data Types

Demand depends on predictable calendar factors such as the season, day–of–the–week and time–of–day considerations. We grouped separately Mondays, Tuesdays through Fridays, Saturdays, and Sundays, as well as holidays. In contrast to all of the earlier work on this problem, we ignored seasonal considerations and let the network and training algorithm adjust as needed. The advantage of this was the ability to form larger training data sets. We thus constructed twelve networks, one

| type | m(t+1) | e(t) | d(t+1) |
|------|--------|------|--------|
| Monday | m(t-3) | m(t) | d(t-3) |
| Tue.-Fri. | m(t-1) | m(t) | d(t-1) |
| Saturday | m(t-1) | m(t-1) | d(t-1) |
| Sunday | m(t-2) | m(t-2) | d(t- 2) |

Table 1: Most recent peaks of a two-feature set

| type | m(t+1) | | | e(t) | | | d(t+1) | | |
|------|-----|-------|------|-----|-------|------|-----|-------|------|
|  | LLS | LOESS | BP | LLS | LOESS | BP | LLS | LOESS | BP |
| Monday | 3.78 | 2.45 | 2.42 | 1.73 | 2.43 | 1.59 | 4.40 | 3.30 | 2.69 |
| Tue.-Fri. | 3.01 | 2.44 | 1.98 | 1.89 | 3.04 | 1.65 | 3.29 | 3.81 | 2.49 |
| Saturday | 3.37 | 2.60 | 2.36 | 4.54 | 3.76 | 3.10 | 3.48 | 3.25 | 2.06 |
| Sunday | 4.83 | 3.28 | 3.79 | 4.89 | 2.74 | 3.81 | 4.26 | 2.44 | 3.03 |

Table 2: Forecasting accuracies (percentage absolute error) for three basic methods

for each pair consisting of one of these four types of days and one of the three daily extremes to be forecast. Demand also depends heavily upon weather which is the primary random factor affecting forecasts. This dependency can be seen in the scatter plots of current demand vs. previous demand and temperature in Figure 2, particularly in the projection onto the 'current demand–temperature' plane which shows a pronounced "U"-shaped nonlinearity. A two-feature set consisting of the most recent peaks and average temperatures over the three cities and the preceding six hours is employed for testing all three models (Table 1).

## 2.3  Benchmark Results

The three basic forecasting models using the two-featured set are:

1) linear regression model fitted to the data in Figure 2;

2) demand vs. temperature models which roughly model the "U-shaped" nonlinear relationship, (LOESS with .5 span was employed for scatter plot smoothing);

3) backpropagation trained neural networks using 5 logistic nodes in a single hidden layer.

The test set errors are given in Table 2. Note that among these three models, BP-trained neural networks gives superior test set performance on all but Sundays. These models all give results comparable to those obtained in our earlier work on forecasting demands for Tuesday-Friday using autoregressive neural networks (Yuan and Fine [1992]).

# 3  Projection Pursuit Training

Satisfactory forecasting performance of the neural networks described above relies on the appropriate choice of feature sets and network architectures. Unfortunately, BP can only address the problem of appropriate architecture and relevant feature sets through repeated time–consuming experiments. Modeling of high–dimensional input features using gradient search requires extensive computation. We were thus prompted to look at other network structures and at training algorithms that could make it easier to explore architecture and training problems. Our initial attempt combined the dimension reduction algorithm cf SIR (Li [1991]), currently replaced by an algorithm of our devising sketched in Section 4, and backfitting (Hastie et.al [1990]) to implement a neural network version of projection pursuit regression (PPR).

## 3.1  The Algorithm

A general nonlinear regression model for a forecast variable $y$ in terms of a vector $x$ of input variables and model noise $\epsilon$, independent of $x$, is given by

$$y = f(\beta_1' x, \beta_2' x, .., \beta_k' x, \epsilon) \qquad (*).$$

A least mean square predictor is the conditional expectation $E(y|x)$. The projection pursuit model/approximation of this conditional expectation is given in terms of a family of SISO functions $\Xi_1, \Xi_2, .., \Xi_k$ by

$$E(y|x) = \sum_{i=1}^{k} \Xi_i(\beta_i' x) + \beta_0.$$

A single hidden layer neural network can approximate this representation by introducing subnets whose summed outputs approximate the individual $\Xi_j$.

We train such a 'projection pursuit network' with nodes partitioned into subnets, representing the $\Xi_i$, by training the subnets individually in rotation. In this we follow the statistical regression notion of backfitting. The subnet $\Xi_i$ is trained to predict the residuals resulting from the difference between the weighted outputs of the other $k-1$ subnets and the true value of the demand variable. After a number of training cycles one then proceeds to the next subnet and repeats the process. The inputs to each subnet $\Xi_i$ are the low–dimensional projections $\beta_i' x$ of the regression model. One termination criteria for determining the number of subnets $k$ is to stop adding subnets when the projection appears to be normally distributed; results of Diaconis and Freedman point out that 'most' projections will be so distributed and thus are 'uninteresting'. The directions $\beta_i$ can be found by minimizing some projection index which defines interesting projections that deviates from Gaussian distributions (e.g., Friedman [1987]). Each $\beta_i$ determines the weights connecting the input to subnet $\Xi_i$. The whole projection pursuit regression process is simplified by decoupling the direction $\beta$ search from training the SISO subnets. Albeit, its success depends upon an ability to rapidly discern the significant directions $\beta_i$.

## 3.2  Implementations

There are several variants in the implementation of projection pursuit training algorithms. General PPR procedure can be implemented in one stage by computa-

| type | m(t+1) | e(t) | d(t+1) |
|------|--------|------|--------|
| Monday | 2.35/3.45 | 1.25/1.60 | 2.76/3.49 |
| Tue.-Fri. | 2.37/2.83 | 1.65/1.66 | 2.15/2.66 |
| Saturday | 2.67/3.16 | 2.78/3.96 | 2.57/3.04 |
| Sunday | 3.15/5.38 | 2.63/3.67 | 2.29/3.61 |

Table 3: Forecasting performance (training/testing percentage error) of projection pursuit trained networks

tionally intensive numerical methods, or in a two–stage heuristic (finding $\beta_i$, then $\Xi_i$) as proposed here. It can be implemented with or without backfitting after the PPR phase is done. Intrator [1992] has recently suggested incorporating the projection index into the objective function and then running an overall BPA. Other variants in training each $\Xi_i$ net include using nonparametric smoothing techniques such as LOESS or kernel methods. BP training can then be applied only in the last stage to fit the smoothed curves so obtained. The complexity of each subnet is then largely determined by the smoothing parameters, like window sizes, inherent in most nonparametric smoothing techniques. Another practical advantage of this process is that one can incorporate easily fixed functions of a single variable ( e.g. linear nodes or quadratic nodes ) when one's prior knowledge of the data source suggests that such components may be present. Our current implementation employs the two-stage algorithm with simple (either one or two nodes) logistic $\Xi_i$ subnets. Each SISO $\Xi_i$ net runs a BP algorithm to fit the data. The directions $\beta_i$ are calculated based on minimizing a projection index (dispersion of level–sets, described in Section 4) which can be executed in a direct fashion. One can encourage the convergence of backfitting by using a relaxation parameter (like a momentum parameter in BPA ) to control the amount updated in the current direction. Training (fitting) of each (SISO) $\Xi_i$ net can be carried out more efficiently than running BP based on high–dimensional inputs, for example, it is less expensive to evaluate the Hessian matrices in a $\Xi_i$ net than in a full BPA networks.

## 3.3   Forecasting Results

Experimental results were obtained using the two-component feature data sets which gave the earlier baseline performance. To calibrate the performance we employed in all twelve projection pursuit trained networks an uniform architecture of three subnets ( a $(1, 2, 2)$-logistic network), matching the 5 nodes of the BP network of Section 2.The number of backfitting cycles was set to 20 with a relaxation parameter $\omega = 0.1$. BPA was employed for fitting each $\Xi_i$net. The training/testing percentage absolute errors are given in Table 3. The limited data sets in the cases of individual days (Monday, Saturday, Sunday) led to failure in generalization that could have been prevented by using one or two, rather than three, subnets.

# 4  Dimension Reduction

## 4.1  Index of Level–Set Dispersion

A key step in the projection pursuit training algorithm is to find for each $\Xi_i$ net the projection direction $\beta_i$, an instance of the important problem of economically choosing input features/variables in constructing a forecasting model. In general, the fewer the number of input features, the more likely are the results to generalize from training set performance to test set performance- reduction in variance at the possible expense of increase in bias. Our controlled size subnet projection pursuit training algorithm deals with part of the complexity problem, provided that the input features are fixed. We turn now to our approach to finding input features or search directions based on minimizing an index of dispersion of level–sets. Li [1991] proposed taking an inverse ('slicing the y's') point of view to estimate the directions $\beta_i$. The justification provided for this so–called slicing inverse regression (SIR) method, however, requires that the input or feature vector $x$ be elliptically symmetrically distributed, and this is not likely to be the case in our electric load forecasting problem. The basic idea behind minimizing dispersion of level–sets is that from Eq. (*) we see that a fixed value of $y$, and small noise $\epsilon$, implies a highly constrained set of values for $\beta_1'x, ..., \beta_k'x$, while leaving unconstrained the components of $x$ that lie in the subspace $B^\perp$ orthogonal to that space $B$ spanned by the $\beta_i$. Hence, if one has a good number of $i.i.d.$ observations sharing a similar value of the response $y$, then there should be more dispersion of input vectors projected into $B^\perp$ than along the projections into $B$. We implement this by quantizing the observed $y$ values into, say, $H$ slices, with $L_h$ denoting the $h$ th level–set containing those inputs with y-value in the $h$ th slice, and $\bar{x_h}$ is their sample mean. The $\beta$ are then picked as the the eigenvector associated with the smallest eigenvalue of the centered covariance matrix:

$$\sum_{h=1}^{H} \sum_{x_i \in L_h} (x_i - \bar{x_h})(x_i - \bar{x_h})'.$$

## 4.2  Implementations

In practical implementations, one may discard both extremes of the family of $H$ level sets (trimming) to avoid large response values when it is believed that they may correspond to large magnitudes of input components. One should also standardize initially the input data to a unit sample covariance matrix. Otherwise, our results will reflect the distribution of $x$ rather than the functional relationship of Eq. (*). We have applied this projection index both in finding the $\beta_i$ during projection pursuit training and in reducing a high–dimensional feature set to a low–dimensional feature set. We have implemented such a feature selection scheme for forecasting the Monday - Friday evening peaks.The initial feature set consists of thirteen hourly loads from 1am to 1pm, thirteen hourly temperatures from 1am to 1pm and the temperature around the peak times. Three eigenvectors of the centered covariance matrix were chosen, thereby reducing a 27–dimensional feature set to a 3–dimensional one. We then ran a standard BPA on this reduced featured set and tested on the 1991 data. We obtained a percentage absolute error of 1.6% (rms error about 100 MWH), which is as good as all of our previous efforts.

## Acknowledgements

Partial support for this research was provided by NSF Grant No. ECS–9017493.

We wish to thank Prof. J. Hwang, Mathematics Department, Cornell, for initial discussions of SIR and are grateful to Dr. P.D. Yeshakul, American Electric Service Corp., for providing the data set and instructing us patiently in the lore of short–term load forecasting.

## References

Connor, J., L. Atlas, R.D. Martin [1991], Recurrent networks and NARMA modeling, NIPS 91.

Damborg, M., M.El-Sharkawi, R. Marks II [1990], Potential of artificial neural networks in power system operation, *Proc. 1990 IEEE Inter.Symp. on Circuits and Systems*, **4**, 2933–2937.

Friedman, J. [1987], Exploratory projection pursuit, *J. Amer. Stat. Assn.*, **82**, 249-266.

Gross,G., F. Galiana [1987], Short–term load forecasting, *Proc. IEEE*, **75**, 1558–1573.

Hastie, T., R. Tibshirani [1990], *Generalized Additive Models*,Chapman and Hall.

Huber, P. [1985], Projection pursuit, *The Annals of Statistics*,13, 435-475.

Intrator, N. [1992] Combinining exploratory projection pursuit and projection pursuit regression with applicatons to neural networks, To appear in *Neural Computation*.

Li, K.-C. [1991] Slicing inverse regression for dimension reduction, *Journal of American Statistical Assoc.*, **86**.

Willis, H.L., J.F.D. Northcote–Green [1984], Comparison tests of fourteen load forecasting methods, *IEEE Trans. on Power Apparatus and Systems*, **PAS–103**, 1190–1197.

Yuan, J–L., T.L.Fine [1992], Forecasting demand for electric power using autoregressive neural networks, *Proc. Conf. on Info. Sci. and Systems*, Princeton, NJ.